# Learning a Small Mixture of Trees[*]

**M. Pawan Kumar**
Computer Science Department
Stanford University
pawan@cs.stanford.edu

**Daphne Koller**
Computer Science Department
Stanford University
koller@cs.stanford.edu

## Abstract

The problem of approximating a given probability distribution using a simpler distribution plays an important role in several areas of machine learning, for example variational inference and classification. Within this context, we consider the task of learning a mixture of tree distributions. Although mixtures of trees can be learned by minimizing the KL-divergence using an EM algorithm, its success depends heavily on the initialization. We propose an efficient strategy for obtaining a good initial set of trees that attempts to cover the entire observed distribution by minimizing the $\alpha$-divergence with $\alpha = \infty$. We formulate the problem using the fractional covering framework and present a convergent sequential algorithm that only relies on solving a convex program at each iteration. Compared to previous methods, our approach results in a significantly smaller mixture of trees that provides similar or better accuracies. We demonstrate the usefulness of our approach by learning pictorial structures for face recognition.

## 1 Introduction

Probabilistic models provide a powerful and intuitive framework for formulating several problems in machine learning and its application areas, such as computer vision and computational biology. A critical choice to be made when using a probabilistic model is its complexity. For example, consider a system that involves $n$ random variables. A probabilistic model that defines a clique of size $n$ has the ability to model any distribution over these random variables. However, the task of learning and inference on such a model becomes computationally intractable. The other extreme case is to define a tree structured model that allows for efficient learning [3] and inference [23]. However, tree distributions have a restrictive form. Hence, they are not suitable for all applications.

A natural way to alleviate the deficiencies of tree distributions is to use a mixture of trees [21]. Mixtures of trees can be employed as accurate models for several interesting problems such as pose estimation [11] and recognition [5, 12]. In order to facilitate their use, we consider the problem of learning them by approximating an observed distribution. Note that the mixture can be learned by minimizing the Kullback-Leibler (KL) divergence with respect to the observed distribution using an expectation-maximization (EM) algorithm [21]. However, there are two main drawbacks of this approach: (i) minimization of KL divergence mostly tries to explain the dominant mode of the observed distribution [22], that is it does not explain the entire distribution; and (ii) as the EM algorithm is prone to local minima, its success depends heavily on the initialization. An intuitive solution to both these problems is to obtain an initial set of trees that covers as much of the observed distribution as possible. To this end, we pose the learning problem as that of obtaining a set of trees that minimize a suitable $\alpha$-divergence [25].

The $\alpha$-divergence measures are a family of functions over two probability distributions that measure the information gain contained in them: that is, given the first distribution, how much information is obtained by observing the second distribution. They form a complete family of measures, in that no other function satisfies all the postulates of information gain [25]. When used as an objective

---

[*]This work was supported by DARPA SA4996-10929-4 and the Boeing company.

function to approximate an observed distribution, the value of $\alpha$ plays a significant role. For example, when $\alpha = 1$ we obtain the KL divergence. As the value of $\alpha$ keeps increasing, the divergence measure becomes more and more *inclusive* [8], that is it tries to cover as much of the observed distribution as possible [22]. Hence, a natural choice for our task of obtaining a good initial estimate would be to set $\alpha = \infty$.

We formulate the minimization of $\alpha$-divergence with $\alpha = \infty$ within the fractional covering framework [24]. However, the standard iterative algorithm for solving fractional covering is not readily applicable to our problem due to its small stepsize. In order to overcome this deficiency we adapt this approach specifically for the task of learning mixtures of trees. Each iteration of our approach adds one tree to the mixture and only requires solving a convex optimization problem. In practice, our strategy converges within a small number of iterations thereby resulting in a small mixture of trees. We demonstrate the effectiveness of our approach by providing a comparison with state of the art methods and learning pictorial structures [6] for face recognition.

## 2 Related Work

The mixture of trees model was introduced by Meila and Jordan [21] who highlighted its appeal by providing simple inference and sampling algorithms. They also described an EM algorithm that learned a mixture of trees by minimizing the KL divergence. However, the accuracy of the EM algorithm is highly dependent on the initial estimate of the mixture. This is evident in the fact that their experiments required a large mixture of trees to explain the observed distribution, due to random initialization.

Several works have attempted to obtain a good set of trees by devising algorithms for minimizing the KL divergence [8, 13, 19, 26]. In contrast, our method uses $\alpha = \infty$, thereby providing a set of trees that covers the entire observed distribution. It has been shown that mixture of trees admit a decomposable prior [20]. In other words, one can concisely specify a certain prior probability for each of the exponential number of tree structures for a given set of random variables. Kirschner and Smyth [14] have also proposed a method to handle a countably infinite mixture of trees. However, the complexity of both learning and inference in these models restricts their practical use.

Researchers have also considered mixtures of trees in the log-probability space. Unlike a mixture in the probability space considered in this paper (which contains a hidden variable), mixtures of trees in log-probability space still define pairwise Markov networks. Such mixtures of trees have been used to obtain upper bounds on the log partition function [27]. However, in this case, the mixture is obtained by considering subgraphs of a given graphical model instead of minimizing a divergence measure with respect to the observed data. Finally, we note that semi-metric distance functions can be approximated to a mixture of tree metrics using the fractional packing framework [24]. This allows us to approximate semi-metric probabilistic models to a simpler mixture of (not necessarily tree) models whose pairwise potentials are defined by tree metrics [15, 17].

## 3 Preliminaries

**Tree Distribution.** Consider a set of $n$ random variables $\mathcal{V} = \{v_1, \cdots, v_n\}$, where each variable $v_a \in \mathcal{X}_a$. We represent a labeling of the random variables (i.e. a particular assignment of values) as a vector $\mathbf{x} = \{x_a | a = 1, \cdots, n\}$. A tree structured model defined over the random variables $\mathcal{V}$ is a graph whose nodes correspond to the random variables and whose edges $\mathcal{E}$ define a tree. Such a model assigns a probability to each labeling that can be written as

$$\Pr(\mathbf{x}|\boldsymbol{\theta}^T) = \frac{1}{Z(\boldsymbol{\theta}^T)} \frac{\prod_{(v_a, v_b) \in \mathcal{E}} \theta_{ab}^T(x_a, x_b)}{\prod_{v_a \in \mathcal{V}} \theta_a^T(x_a)^{deg(a)-1}}. \tag{1}$$

Here $\theta_a^T(\cdot)$ refers to unary potentials whose values depend on one variable at a time, and $\theta_{ab}^T(\cdot, \cdot)$ refers to pairwise potentials whose values depend on two neighboring variables at a time. The vector $\boldsymbol{\theta}^T$ is the parameter of the model (which consists of all the potentials) and $Z(\boldsymbol{\theta}^T)$ is the partition function which ensures that the probability sums to one. The term $deg(a)$ denotes the degree of the variable $v_a$.

**Mixture of Trees.** As the name suggests, a mixture of trees is defined by a set of trees along with a probability distribution over them, that is $\boldsymbol{\theta}^M = \{(\boldsymbol{\theta}^T, \rho^T)\}$ such that mixture coefficients $\rho^T > 0$ for all $T$ and $\sum_T \rho^T = 1$. It defines the probability of a given labeling as

$$\Pr(\mathbf{x}|\boldsymbol{\theta}^M) = \sum_T \rho^T \Pr(\mathbf{x}|\boldsymbol{\theta}^T). \tag{2}$$

$\alpha$**-Divergence.** The $\alpha$-divergence between distributions $\Pr(\cdot|\boldsymbol{\theta}^1)$ (say the observed distribution) and $\Pr(\cdot|\boldsymbol{\theta}^2)$ (the simpler distribution) is given by

$$D_\alpha(\boldsymbol{\theta}^1||\boldsymbol{\theta}^2) = \frac{1}{\alpha-1}\log\left(\sum_{\mathbf{x}}\frac{\Pr(\mathbf{x}|\boldsymbol{\theta}^1)^\alpha}{\Pr(\mathbf{x}|\boldsymbol{\theta}^2)^{\alpha-1}}\right). \tag{3}$$

The $\alpha$-divergence measure is strictly non-negative and is equal to $0$ if and only if $\boldsymbol{\theta}^1$ is a reparameterization of $\boldsymbol{\theta}^2$. It is a generalization of KL divergence which corresponds to $\alpha=1$, that is

$$D_1(\boldsymbol{\theta}^1||\boldsymbol{\theta}^2) = \sum_{\mathbf{x}}\Pr(\mathbf{x}|\boldsymbol{\theta}^1)\log\frac{\Pr(\mathbf{x}|\boldsymbol{\theta}^1)}{\Pr(\mathbf{x}|\boldsymbol{\theta}^2)}. \tag{4}$$

As mentioned earlier, we are interested in the case where $\alpha=\infty$, that is

$$D_\infty(\boldsymbol{\theta}^1||\boldsymbol{\theta}^2) = \max_{\mathbf{x}}\log\frac{\Pr(\mathbf{x}|\boldsymbol{\theta}^1)}{\Pr(\mathbf{x}|\boldsymbol{\theta}^2)}. \tag{5}$$

The inclusive property of $\alpha=\infty$ is evident from the above formula. Since we would like to minimize the maximum ratio of probabilities (i.e. the worst case), we need to ensure that no value of $\Pr(\mathbf{x}|\boldsymbol{\theta}^2)$ is very small, that is the entire distribution is covered. In contrast, the KL divergence can admit very small values of $\Pr(\mathbf{x}|\boldsymbol{\theta}^2)$ since it is concerned with the summation shown in equation (4) (and not the worst case). To avoid confusion, we shall refer to the case where $\alpha=1$ as KL divergence and the $\alpha=\infty$ case as $\alpha$-divergence throughout this paper.

**The Learning Problem.** Given a set of samples $\{\mathbf{x}_i, i=1,\cdots,m\}$ along with their probabilities $\hat{P}(\mathbf{x}_i)$, our task is to learn a mixture of trees $\boldsymbol{\theta}^{M^*}$ such that

$$\boldsymbol{\theta}^{M^*} = \arg\min_{\boldsymbol{\theta}^M}\left(\max_i\log\frac{\hat{P}(\mathbf{x}_i)}{\Pr(\mathbf{x}_i|\boldsymbol{\theta}^M)}\right) = \arg\max_{\boldsymbol{\theta}^M}\left(\min_i\frac{\Pr(\mathbf{x}_i|\boldsymbol{\theta}^M)}{\hat{P}(\mathbf{x}_i)}\right). \tag{6}$$

We will concentrate on the second form in the above equation (where the logarithm has been dropped). We define $\mathcal{T} = \{\boldsymbol{\theta}^{T_j}\}$ to be the set of all $t$ tree distributions that are defined over $n$ variables. It follows that the probability of a labeling for any mixture of trees can be written as

$$\Pr(\mathbf{x}|\boldsymbol{\theta}^M) = \sum_j \rho_j\Pr(\mathbf{x}|\boldsymbol{\theta}^{T_j}), \tag{7}$$

for suitable values of $\rho_j$. Note that the mixing coefficients $\boldsymbol{\rho}$ should define a valid probability distribution. In other words, $\boldsymbol{\rho}$ belongs to the polytope $\mathcal{P}$ defined as

$$\boldsymbol{\rho}\in\mathcal{P}\Rightarrow\sum_j\rho_j=1,\rho_j\geq 0, \forall j=1,\cdots,t. \tag{8}$$

Our task is to find a sparse vector $\boldsymbol{\rho}$ that minimizes the $\alpha$-divergence with respect to the observed distribution. In order to formally specify the minimization of $\alpha$-divergence as an optimization problem, we define an $m\times t$ matrix $\mathbf{A}$ and an $m\times 1$ vector $\mathbf{b}$ such that

$$A(i,j) = \Pr(\mathbf{x}_i|\boldsymbol{\theta}^{T_j}) \text{ and } b_i = \hat{P}(\mathbf{x}_i). \tag{9}$$

We denote the $i^{th}$ row of $\mathbf{A}$ as $\mathbf{a}_i$ and the $i^{th}$ element of $\mathbf{b}$ as $b_i$. Using the above notation, the learning problem can be specified as

$$\max_{\boldsymbol{\rho}}\lambda_{\boldsymbol{\rho}},$$
$$\text{s.t.}\quad \mathbf{a}_i\boldsymbol{\rho}\geq\lambda_{\boldsymbol{\rho}}b_i,\forall i$$
$$\boldsymbol{\rho}\in\mathcal{P}, \tag{10}$$

where $\lambda_{\boldsymbol{\rho}} = \min_i\mathbf{a}_i\boldsymbol{\rho}/b_i$ due to the form of the above LP. The above formulation suggests that a natural way to attack the problem would be to use the fractional covering framework [24]. We begin by briefly describing fractional covering in the next section.

## 4 Fractional Covering

Given an $m \times t$ matrix $\mathbf{A}$ and an $m \times 1$ vector $\mathbf{b} > 0$, the fractional covering problem is to determine whether there exists a vector $\boldsymbol{\rho} \in \mathcal{P}$ such that $\mathbf{A}\boldsymbol{\rho} \geq \mathbf{b}$. The only restriction on the polytope $\mathcal{P}$ is that $\mathbf{A}\boldsymbol{\rho} \geq 0$ for all $\boldsymbol{\rho} \in \mathcal{P}$, which is clearly satisfied by our learning problem (since $\mathbf{a}_i\boldsymbol{\rho}$ is the probability of $\mathbf{x}_i$ specified by the mixture of trees corresponding to $\boldsymbol{\rho}$). Let

$$\lambda^* = \max_{\boldsymbol{\rho}} \min_i \frac{\mathbf{a}_i\boldsymbol{\rho}}{b_i}. \tag{11}$$

If $\lambda^* < 1$ then clearly there does not exist a $\boldsymbol{\rho}$ such that $\mathbf{A}\boldsymbol{\rho} \geq \mathbf{b}$. However, if $\lambda^* \geq 1$, then the fractional covering problem requires us to find an $\epsilon$-optimal solution, that is find a $\boldsymbol{\rho}$ such that

$$\mathbf{A}\boldsymbol{\rho} \geq (1 - \epsilon)\lambda^*\mathbf{b}, \tag{12}$$

where $\epsilon > 0$ is a user-specified tolerance factor. Using the definitions of $\mathbf{A}$, $\mathbf{b}$ and $\boldsymbol{\rho}$ from the previous section, we observe that in our case $\lambda^* = 1$. In other words, there exists a solution such that $\mathbf{A}\boldsymbol{\rho} = \mathbf{b}$. This can easily be seen by considering a tree with parameter $\boldsymbol{\theta}^{T_j}$ such that

$$\Pr(\mathbf{x}_i|\boldsymbol{\theta}^{T_j}) = \begin{cases} 1 & \text{if} \quad i = j, \\ 0 & \text{otherwise}, \end{cases} \tag{13}$$

and setting $\rho_j = \hat{P}(\mathbf{x}_j)$. The above solution provides an $\alpha$-divergence of 0 but at the cost of introducing $m$ trees in the mixture (where $m$ is the number of samples provided). We would like to find an $\epsilon$-optimal solution with a smaller number of trees by solving the LP (10). However, we cannot employ standard interior point algorithms for optimizing problem (10). This is due to the fact that each of its $m$ constraints is defined over an infinite number of unknowns (specifically, the mixture coefficients for each of the infinite number of tree distributions defined over the $n$ random variables). Fortunately, Plotkin *et al.* [24] provide an iterative algorithm for solving problem (10) that can handle arbitrarily large number of unknowns in every constraint.

**The Fractional Covering Algorithm.** In order to obtain a solution to problem (10), we solve the following related problem:

$$\min_{\boldsymbol{\rho} \in \mathcal{P}} \Phi(\mathbf{y}) \equiv \mathbf{y}^\top \mathbf{b},$$

$$\text{s.t.} \quad y_i = \frac{1}{b_i} \exp\left(-\beta \frac{\mathbf{a}_i\boldsymbol{\rho}}{b_i}\right). \tag{14}$$

The objective function $\Phi(\mathbf{y})$ is called the potential function for fractional covering. Plotkin *et al.* [24] showed that minimizing $\Phi(\mathbf{y})$ solves the original fractional covering problem. The term $\beta$ is a parameter that is inversely proportional to the stepsize $\sigma$ of the algorithm. The fractional covering algorithm is an iterative strategy. At iteration $t$, the variable $\boldsymbol{\rho}^t$ is updated as $\boldsymbol{\rho}^t \leftarrow (1-\sigma)\boldsymbol{\rho}^{t-1} + \sigma\boldsymbol{\rho}'$ such that the update attempts to decrease the potential function. Specifically, the algorithm proposed in [24] suggests using the first order approximation of $\Phi(\mathbf{y})$, that is

$$\boldsymbol{\rho}' = \arg\min_{\boldsymbol{\rho}} \left( \sum_i y_i'(b_i - \beta\sigma\mathbf{a}_i\boldsymbol{\rho}) \right) = \arg\max_{\boldsymbol{\rho}} \mathbf{y}'^\top \mathbf{A}\boldsymbol{\rho}. \tag{15}$$

where

$$y_i' = \frac{1}{b_i} \exp\left(-\beta \frac{(1-\sigma)\mathbf{a}_i\boldsymbol{\rho}}{b_i}\right). \tag{16}$$

Typically, the above problem is easy to solve (including for our case, as will be seen in the next section). Furthermore, for a sufficiently large value of $\beta$ ($\propto \log m$) the above update rule decreases $\Phi(\mathbf{y})$. In more detail, the algorithm of [24] is as follows:

- Define $w = \max_{\boldsymbol{\rho}} \max_i \mathbf{a}_i\boldsymbol{\rho}/b_i$ to be the *width* of the problem.
- Start with an initial solution $\boldsymbol{\rho}_0$.
- Define $\lambda_{\boldsymbol{\rho}_0} = \min_i \mathbf{a}_i\boldsymbol{\rho}_0/b_i$, and $\sigma = \epsilon/(4\beta w)$.
- While $\lambda_{\boldsymbol{\rho}} < 2\lambda_{\boldsymbol{\rho}_0}$, at iteration $t$:
    - Define $\mathbf{y}'$ as shown in equation (16).
    - Find $\boldsymbol{\rho}' = \arg\max_{\boldsymbol{\rho} \in \mathcal{P}} \mathbf{y}'^\top \mathbf{A}\boldsymbol{\rho}$.
    - Update $\boldsymbol{\rho}^t \leftarrow (1-\sigma)\boldsymbol{\rho}^{t-1} + \sigma\boldsymbol{\rho}^*$.

Plotkin *et al.* [24] suggest starting with a tolerance factor of $\epsilon_0 = 1/6$ and dividing the value of $\epsilon_0$ by 2 after every call to the above procedure terminates. This process is continued until a sufficiently accurate (i.e. an $\epsilon$-optimal) solution is recovered. Note that during each call to the above procedure the potential function $\Phi(\mathbf{y})$ is both upper and lower bounded, specifically

$$\exp(-2\beta\lambda_{\boldsymbol{\rho}_0}) \leq \Phi(\mathbf{y}) \leq m \exp(-\beta\lambda_{\boldsymbol{\rho}_0}). \tag{17}$$

Furthermore, we are guaranteed to decrease the value of $\Phi(\mathbf{y})$ at each iteration. Hence, it follows that the above algorithm will converge. We refer the reader to [24] for more details.

## 5 Modifying Fractional Covering

The above algorithm provides an elegant way to solve the general fractional covering problem. However, as will be seen shortly, in our case it leads to undesirable solutions. Nevertheless, we show that appropriate modifications can be made to obtain a small and accurate mixture of trees. We begin by identify the deficiencies of the fractional covering algorithm for our learning problem.

### 5.1 Drawbacks of the Algorithm

There are two main drawbacks of fractional covering. First, the value of $\beta$ is typically very large, which results in a small stepsize $\sigma$. In our experiments, $\beta$ was of the order of $10^3$, which resulted in slow convergence of the algorithm. Second, the update step provides *singleton* trees, that is trees with a probability of 1 for one labeling and 0 for all others. This is due to the fact that, in our case, the update step solves the following problem:

$$\max_{\boldsymbol{\rho}\in\mathcal{P}} \sum_j \left( \sum_i y_i' \rho_j \Pr(\mathbf{x}_i|\boldsymbol{\theta}^{T_j}) \right). \tag{18}$$

Note that the above problem is an LP in $\boldsymbol{\rho}$. Hence, there must exist an optimal solution on the vertex on the polytope $\mathcal{P}$. In other words, we obtain a single tree distribution $\boldsymbol{\theta}^{T^*}$ such that

$$\boldsymbol{\theta}^{T^*} = \arg\max_{\boldsymbol{\theta}^T} \left( \sum_i y_i' \Pr(\mathbf{x}_i|\boldsymbol{\theta}^T) \right). \tag{19}$$

The optimal tree distribution for the above problem concentrates the entire mass on the sample $\mathbf{x}_{i'}$ where $i' = \arg\max_i y_i'$. Such singleton trees are not desirable as they also result in slow convergence of the algorithm. Furthermore, the learned mixture only provides a non-zero probability for the samples used during training. Hence, the mixture cannot be used for previously unseen samples, thereby rendering it practically useless. Note that the method of Rosset and Segal [26] also faces a similar problem during their update steps for minimizing the KL divergence. In order to overcome this difficulty, they suggest approximating problem (18) by

$$\boldsymbol{\theta}^{T^*} = \arg\max_{\boldsymbol{\theta}^T} \sum_i y_i' \log\left(\Pr(\mathbf{x}_i|\boldsymbol{\theta}^T)\right), \tag{20}$$

which can be solved efficiently using the Chow-Liu algorithm [3]. However, our preliminary experiments (accuracies not reported) indicate that this approach does not work well for minimizing the potential function $\Phi(\mathbf{y})$.

### 5.2 Fixing the Drawbacks

We adapt the original fractional covering algorithm for our problem in order to overcome the drawbacks mentioned above. The first drawback is handled easily. We start with a small value of $\beta$ and increase it by a factor of 2 if we are not able to reduce the potential function $\Phi(\mathbf{y})$ at a given iteration. Since we are assured that the value of $\Phi(\mathbf{y})$ decreases for a finite value of $\beta$, this procedure is guaranteed to terminate. In our experiments, we initialized $\beta = 1/w$ and its value never exceeded $32/w$. Note that choosing $\beta$ to be inversely proportional to $w$ ensures that the initial values of $y_i'$ in equation (16) are sufficiently large (at least $\exp(-(1-\sigma))$).

In order to address the second drawback, we note that our aim at an iteration $t$ of the algorithm is to reduce the potential function $\Phi(\mathbf{y})$. That is, given the current distribution parameterized by $\boldsymbol{\theta}^{M_t}$ we would like to add a new tree $\boldsymbol{\theta}^{T_t}$ to the mixture that solves the following problem:

$$\boldsymbol{\theta}^{T_t} = \arg\min_{\boldsymbol{\theta}^T} \left[ \Phi(\mathbf{y}) \equiv \sum_i y_i' \exp\left( -\beta \frac{\sigma \Pr(\mathbf{x}_i|\boldsymbol{\theta}^T)}{\hat{P}(\mathbf{x}_i)} \right) \right] \tag{21}$$

$$\text{s.t.} \quad \sum_i \Pr(\mathbf{x}_i | \boldsymbol{\theta}^T) \leq 1, \;\; \Pr(\mathbf{x}_i | \boldsymbol{\theta}^T) \geq 0, \forall i = 1, \cdots, m, \tag{22}$$

$$\boldsymbol{\theta}^T \in \mathcal{T}. \tag{23}$$

Here, $\mathcal{T}$ is the set of all tree distributions defined over $n$ random variables. Note that the algorithm of [24] optimizes the first order approximation of the objective function (21). However, as seen previously, for our problem this results in an undesirable solution. Instead, we directly optimize $\Phi(\mathbf{y})$ using an alternative two step strategy. In the first step, we drop the last constraint from the above problem. In other words, we obtain the values of $\Pr(\mathbf{x}_i | \boldsymbol{\theta}^T)$ that form a valid (but not necessarily tree-structured) distribution and minimize the function $\Phi(\mathbf{y})$. Note that since the $\Phi(\mathbf{y})$ is not linear in $\Pr(\mathbf{x}_i | \boldsymbol{\theta}^T)$, the optimal solution provides a dense distribution $\Pr(\cdot | \boldsymbol{\theta}^T)$ (as opposed to the first order linear approximation which provides a singleton distribution). In the second step, we project these values to a tree distribution. It is easy to see that dropping constraint (23) results in a convex relaxation of the original problem. We solve the convex relaxation using a log-barrier method [1]. Briefly, this implies solving a series of unconstrained optimization problems until we are within a user-specified tolerance value of $\tau$ from the optimal solution. Specifically,

- Set $f = 1$.
- Solve $\min_{\Pr(\cdot | \boldsymbol{\theta}^T)} \left( f\Phi(\mathbf{y}) - \sum_i \log(\Pr(\mathbf{x}_i | \boldsymbol{\theta}^T)) - \log(1 - \sum_i \Pr(\mathbf{x}_i | \boldsymbol{\theta}^T)) \right)$.
- If $m/f \leq \tau$, then stop. Otherwise, update $f = \mu f$ and repeat the previous step.

We used $\mu = 1.5$ in all our experiments, which was sufficient to obtain accurate solutions for the convex relaxation. At each iteration, the unconstrained optimization problem is solved using Newton's method. Recall that Newton's method minimizes a function $g(\mathbf{z})$ by updating the current solution as

$$g(\mathbf{z}) \leftarrow g(\mathbf{z}) - \left( \nabla^2 g(\mathbf{z}) \right)^{-1} \nabla g(\mathbf{z}), \tag{24}$$

where $\nabla^2 g(\cdot)$ denotes the Hessian matrix and $\nabla g(\cdot)$ denotes the gradient vector. Note that the most expensive step in the above approach is the inversion of the Hessian matrix. However, it is easy to verify that in our case all the off-diagonal elements of the Hessian are equal to each other. By taking advantage of this special form of the Hessian, we compute its inverse in $O(m^2)$ time using Gaussian elimination (i.e. linear in the number of elements of the Hessian).

Once the values of $\Pr(\mathbf{x}_i | \boldsymbol{\theta}^T)$ are computed in this manner, they are projected to a tree distribution using the Chow-Liu algorithm [3]. Note that after the projection step we are no longer guaranteed to decrease the function $\Phi(\mathbf{y})$. This would imply that the overall algorithm would not be guaranteed to converge. In order to overcome this problem, if we are unable to decrease $\Phi(\mathbf{y})$ then we determine the sample $\mathbf{x}_{i'}$ such that

$$i' = \arg\max_i \frac{\Pr(\mathbf{x}_i | \boldsymbol{\theta}^{M_t})}{\hat{P}(\mathbf{x}_i)}, \tag{25}$$

that is the sample best explained by the current mixture. We enforce $\Pr(\mathbf{x}_{i'} | \boldsymbol{\theta}^T) = 0$ and solve the above convex relaxation again. Note that the solution to the new convex relaxation (i.e. the one with the newly introduced constraint for sample $\mathbf{x}_{i'}$) can easily be obtained from the solution of the previous convex relaxation using the following update:

$$\Pr(\mathbf{x}_i | \boldsymbol{\theta}^T) \leftarrow \begin{cases} \Pr(\mathbf{x}_i | \boldsymbol{\theta}^T) + \hat{P}(\mathbf{x}_i) \Pr(\mathbf{x}_{i'} | \boldsymbol{\theta}^T)/s & \text{if} \quad i \neq i', \\ 0 & \text{otherwise,} \end{cases} \tag{26}$$

where $s = \sum_i \hat{P}(\mathbf{x}_i)$. In other words, we do not need to use the log-barrier method to solve the new convex relaxation. We then project the updated values of $\Pr(\mathbf{x}_i | \boldsymbol{\theta}^T)$ to a tree distribution. This process of eliminating one sample and projecting to a tree is repeated until we are able to reduce the value of $\Phi(\mathbf{y})$. Note that in the worst case we will eliminate all but one sample (specifically, the one that corresponds to the update scheme of [24]). In other words, we will add a singleton tree. However, in practice our algorithm converges in a small number ($\ll m$) of iterations and provides an accurate mixture of trees. In fact, in all our experiments we never obtained any singleton trees. We conclude the description of our method by noting that once the new tree distribution $\boldsymbol{\theta}^{T_t}$ is obtained, the value of $\sigma$ is easily updated as $\sigma = \arg\min_\sigma \Phi(\mathbf{y})$.

## 6  Experiments

We present a comparison of our method with the state of the art algorithms. We also use it to learn pictorial structures for face recognition. Note that our method is efficient in practice due to the

| Dataset | TANB | MF | Tree | MT | [26] + MT | Our + MT |
|---------|------|----|------|----|-----------|----------|
| Agaricus | **100.0 ± 0** | 99.45 ± 0.004 | 98.65 ± 0.32 | **99.98 ± 0.04** | **100.0 ± 0** | **100.0 ± 0** |
| Nursery | 93.0 ± 0 | 98.0 ± 0.01 | 92.17 ± 0.38 | **99.2 ± 0.02** | 98.35 ± 0.30 | **99.28 ± 0.13** |
| Splice | 94.9 ± 0.9 | - | **95.7 ± 0.2** | 95.5 ± 0.3 | 95.6 ± 0.42 | **96.1 ± 0.15** |

Table 1: *Classification accuracies for the datasets used in [21]. The first column shows the name of the dataset. The subsequent columns show the mean accuracies and the standard deviation over 5 trials of tree-augmented naive Bayes [10], mixture of factorial distributions [2], single tree classifier [3], mixture of trees with random initialization (i.e. the numbers reported in [21]), initialization with [26] and initialization with our approach. Note that our method provides similar accuracies to [21] while using a smaller mixture of trees (see text).*

special form of the Hessian matrix (for the log-barrier method) and the Chow-Liu algorithm [3, 21] (for the projection to tree distributions). In all our experiments, each iteration takes only 5 to 10 minutes (and the number of iterations is equal to the number of trees in the mixture).

**Comparison with Previous Work.** As mentioned earlier, our approach can be used to obtain a good initialization for the EM algorithm of [21] since it minimizes $\alpha$-divergence (providing complementary information to the KL-divergence used in [21]). This is in contrast to the random initializations used in the experiments of [21] or the initialization obtained by [26] (that also attempts to minimize the KL-divergence). We consider the task of using the mixture of trees as a classifier, that is given training data that consists of feature vectors $\mathbf{x}_i$ together with the class values $c_i$, the task is to correctly classify previously unseen test feature vectors. Following the protocol of [21], this can be achieved in two ways. For the first type of classifier, we append the feature vector $\mathbf{x}_i$ with its class value $c_i$ to obtain a new feature vector $\mathbf{x}_i'$. We then learn a mixture of tree that predicts the probability of $\mathbf{x}_i'$. Given a new feature vector $\mathbf{x}$ we assign it the class $c$ that results in the highest probability. For the second type of classifier, we learn a mixture of trees for each class value such that it predicts the probability of a feature vector belonging to that particular class. Once again, given a new feature vector $\mathbf{x}$ we assign it the class $c$ which results in the probability.

We tested our approach on the three discrete valued datasets used in [21]. In all our experiments, we initialized the mixture with a single tree obtained from the Chow-Liu algorithm. We closely followed the experimental setup of [21] to ensure that the comparisons are fair. Table 1 provides the accuracy of our approach together with the results reported in [21]. For 'Splice' the first classifier provides the best results, while 'Agaricus' and 'Nursery' use the second classifier. Note that our method provides similar accuracies to [21]. More importantly, it uses a smaller mixture of trees to achieve these results. Specifically, the method of [21] uses 12, 30 and 3 trees for the three datasets respectively. In contrast our method uses 3-5 trees for 'Agaricus', 10-15 trees for 'Nursery' and 2 trees for Splice (where the number of trees in the mixture was obtained using a validation dataset, see [21] for details). Furthermore, unlike [21, 26], we obtain better accuracies by using a mixture of trees instead of a single tree for the 'Splice' dataset. It is worth noting that [26] also provided a small set of initial trees (with comparable size to our method). However, since the trees do not cover the entire observed distribution, their method provides less accurate results.

**Face Recognition.** We tested our approach on the task of recognizing faces using the publicly available dataset[1] containing the faces of 11 characters in an episode of 'Buffy the Vampire Slayer'. The total number of faces in the dataset is 24,244. For each face we are provided with the location of 13 facial features (see Fig. 1). Furthermore, for each facial feature, we are also provided with a vector that represents the appearance of that facial feature [5] (using the normalized grayscale values present in a circular region of radius 7 centered at the facial feature). As noted in previous work [5, 18] the task is challenging due to large intra-class variations in expression and lighting conditions.

Given the appearance vector, the likelihood of each facial feature belonging to a particular character can be found using logistic regression. However, the relative locations of the facial features also offer important cues in distinguishing one character from the other (e.g. the width of the eyes or the distance between an eye and the nose). Typically, in vision systems, this information is not used. In other words, the so-called bag of visual words model is employed. This is due to the somewhat counter-intuitive observation made by several researchers that models that employ spatial prior on the features, e.g. pictorial structures [6], often provide worse recognition accuracies than those that throw away this information. However, this may be due to the fact that often the structure and parameters of pictorial structures and other related models are set by hand.

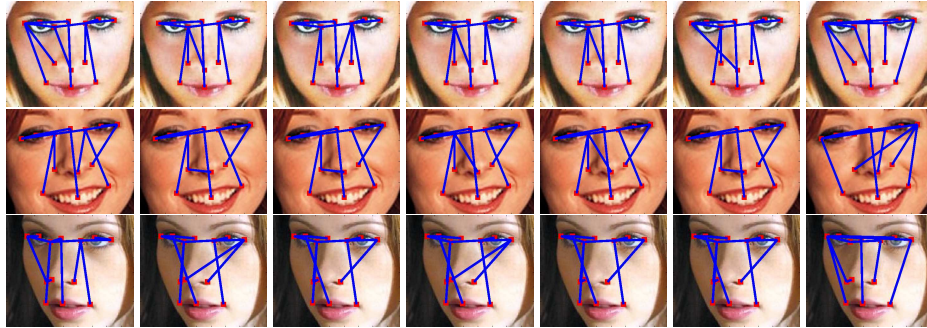

Figure 1: *The structure of the seven trees learned for 3 of the 11 characters using our method. The red squares show the position of the facial features while the blue lines indicate the edges. The structure and parameters of the trees vary significantly, thereby indicating the multimodality of the observed distribution.*

| | 0 | 1 | 2 | 3 | 4 | 5 | 6 | 7 |
|---|---|---|---|---|---|---|---|---|
| [26] | **65.68%** | 66.05% | 66.01% | 66.01% | 66.08% | 66.08% | 66.16% | 66.20% |
| Our | **65.68%** | **66.05%** | **66.65%** | **66.86%** | **67.25%** | **67.48%** | **67.50%** | **67.68%** |

Table 2: *Accuracy for the face recognition experiments. The columns indicate the size of the mixture, ranging from 0 (i.e. the bag of visual words model) to 7 (where the results saturate). Note that our approach, which minimizes the $\alpha$-divergence, provides better results than the method of [26], which minimizes KL-divergence.*

In order to test whether a spatial model can help improve recognition, we learned a mixture of trees for each of the characters. The random variables of the trees correspond to the facial features and their values correspond to the relative location of the facial feature with respect to the center of the nose. The unary potentials of each random variable is specified using the appearance vectors (i.e. the likelihood obtained by logistic regression). In order to obtain the pairwise potentials (i.e. the structure and parameters of the mixture of trees), the faces are normalized to remove global scaling and in-plane rotation using the location of the facial features. We use the faces found in the first 80% of the episode to learn the mixture of trees. The faces found in the remaining 20% of the episode were used as test data. Splitting the dataset in this manner (i.e. a non-random split) ensures that we do not have any trivial cases where a face found in frame $t$ is used for training and a (very similar) face found in frame $t+1$ is used for testing.

Fig. 1 shows the structure of the trees learned for 3 characters. The structures differ significantly between characters, which indicates that different spatial priors are dominant for different characters. Although the structure of the trees for a particular character are similar, they vary considerably in the parameters. This suggests that the distribution is in fact multimodal and therefore cannot be represented accurately using a single tree. Although vision researchers have tried to overcome this problem by using more complex models, e.g. see [4], their use is limited by a lack of efficient learning algorithms. Table 2 shows the accuracy of the mixture of trees learned by the method of [26] and our approach. In this experiment, refining the mixture of trees using the EM algorithm of [21] did not improve the results. This is due to the fact that the training and testing data differ significantly (due to non-random splits, unlike the previous experiments which used random splits of the UCI datasets). In fact, when we split the face dataset randomly, we found that the EM algorithm did help. However, classification problems simulated using random splits of video frames are rare in real-world applications. Since [26] tries to minimize the KL divergence, it mostly tries to explain the dominant mode of the observed distribution. This is evident in the fact that the accuracy of the mixture of trees does not increase significantly as the size of the mixture increases (see table 2, first row). In contrast, the minimization of $\alpha$-divergence provides a diverse set of trees that attempt to explain the entire distribution thereby providing significantly better results (table 2, second row).

## 7   Discussion

We formulated the problem of obtaining a small mixture of trees by minimizing the $\alpha$-divergence within the fractional covering framework. Our experiments indicate that the suitably modified fractional covering algorithm provides accurate models. We believe that our approach offers a natural framework for addressing the problem of minimizing $\alpha$-divergence and could prove useful for other classes of mixture models, for example mixtures of trees in log-probability space for which there exist several efficient and accurate inference algorithms [16, 27]. There also appears to be a connection between fractional covering (proposed in the theory community) and Discrete AdaBoost [7, 9] (proposed in the machine learning community) that merits further exploration.

## Footnotes

[1] Available at http://www.robots.ox.ac.uk/~vgg/research/nface/data.html

# References

[1] S. Boyd and L. Vandenberghe. *Convex Optimization*. Cambridge University Press, 2004.

[2] P. Cheeseman and J. Stutz. Bayesian classification (AutoClass): Theory and results. In *KDD*, pages 153–180, 1995.

[3] C. Chow and C. Liu. Approximating discrete probability distributions with dependence trees. *IEEE Transactions on Information Theory*, 14(3):462–467, 1968.

[4] D. Crandall, P. Felzenszwalb, and D. Huttenlocher. Spatial priors for parts-based recognition using statistical models. In *CVPR*, 2005.

[5] M. Everingham, J. Sivic, and A. Zisserman. Hello! My name is... Buffy - Automatic naming of characters in TV video. In *BMVC*, 2006.

[6] M. Fischler and R. Elschlager. The representation and matching of pictorial structures. *TC*, 22:67–92, January 1973.

[7] Y. Freund and R. Schapire. A decision-theoretic generalization of on-line learning and an application to boosting. *Journal of Computer and System Sciences*, 55(1):119–139, 1997.

[8] B. Frey, R. Patrascu, T. Jaakkola, and J. Moran. Sequentially fitting inclusive trees for inference in noisy-OR networks. In *NIPS*, 2000.

[9] J. Friedman, T. Hastie, and R. Tibshirani. Additive logistic regression: A statistical view of boosting. *Annals of Statistics*, 28(2):337–407, 2000.

[10] N. Friedman, D. Geiger, and M. Goldszmidt. Bayesian network classifiers. *Machine Learning*, 29:131–163, 1997.

[11] S. Ioffe and D. Forsyth. Human tracking with mixtures of trees. In *ICCV*, pages 690–695, 2001.

[12] S. Ioffe and D. Forsyth. Mixtures of trees for object recognition. In *CVPR*, pages 180–185, 2001.

[13] Y. Jing, V. Pavlovic, and J. Rehg. Boosted bayesian network classifiers. *Machine Learning*, 73(2):155–184, 2008.

[14] S. Kirschner and P. Smyth. Infinite mixture of trees. In *ICML*, pages 417–423, 2007.

[15] J. Kleinberg and E. Tardos. Approximation algorithms for classification problems with pairwise relationships: Metric labeling and Markov random fields. In *STOC*, 1999.

[16] V. Kolmogorov. Convergent tree-reweighted message passing for energy minimization. *PAMI*, 2006.

[17] M. P. Kumar and D. Koller. MAP estimation of semi-metric MRFs via hierarchical graph cuts. In *UAI*, 2009.

[18] M. P. Kumar, P. Torr, and A. Zisserman. An invariant large margin nearest neighbour classifier. In *ICCV*, 2007.

[19] Y. Lin, S. Zhu, D. Lee, and B. Taskar. Learning sparse Markov network structure via ensemble-of-trees models. In *AISTATS*, 2009.

[20] M. Meila and T. Jaakkola. Tractable Bayesian learning of tree belief networks. In *UAI*, 2000.

[21] M. Meila and M. Jordan. Learning with a mixture of trees. *JMLR*, 1:1–48, 2000.

[22] T. Minka. Divergence measures and message passing. Technical report, Microsoft Research, 2005.

[23] J. Pearl. *Probabilistic Reasoning in Intelligent Systems: Networks of Plausible Inference*. Morgan-Kauffman, 1988.

[24] S. Plotkin, D. Shmoys, and E. Tardos. Fast approximation algorithms for fractional packing and covering problems. *Mathematics of Operations Research*, 20:257–301, 1995.

[25] A. Renyi. On measures of information and entropy. In *Berkeley Symposium on Mathematics, Statistics and Probability*, pages 547–561, 1961.

[26] S. Rosset and E. Segal. Boosting density estimation. In *NIPS*, 2002.

[27] M. Wainwright, T. Jaakkola, and A. Willsky. A new class of upper bounds on the log partition function. *IEEE Transactions on Information Theory*, 51:2313–2335, 2005.

